# The Entire Regularization Path for the Support Vector Machine

**Trevor Hastie**
Department of Statistics
Stanford University
Stanford, CA 94305, USA
hastie@stanford.edu

**Saharon Rosset**
IBM Watson Research Center
P.O. Box 218
Yorktown Heights, N.Y. 10598
srosset@us.ibm.com

**Robert Tibshirani**
Department of Statistics
Stanford University
Stanford, CA 94305, USA
tibs@stanford.edu

**Ji Zhu**
Department of Statistics
University of Michigan
Ann Arbor, MI 48109-1092
jizhu@umich.edu

## Abstract

In this paper we argue that the choice of the SVM cost parameter can be critical. We then derive an algorithm that can fit the entire path of SVM solutions for every value of the cost parameter, with essentially the same computational cost as fitting one SVM model.

## 1 Introduction

We have a set of $n$ training pairs $x_i, y_i$, where $x_i \in \mathbb{R}^p$ is a $p$-vector of real valued predictors (attributes) for the $i$th observation, $y_i \in \{-1, +1\}$ codes its binary response. The standard criterion for fitting the linear SVM )[1, 2, 3] is

$$\min_{\beta_0, \beta} \frac{1}{2}||\beta||^2 + C \sum_{i=1}^{n} \xi_i, \tag{1}$$

$$\text{subject to, for each } i\text{: } y_i(\beta_0 + x_i^T \beta) \geq 1 - \xi_i.$$

Here the $\xi_i$ are non-negative slack variables that allow points to be on the wrong side of their "soft margin" ($f(x) = \pm 1$), as well as the decision boundary, and $C$ is a cost parameter that controls the amount of overlap. If the data are separable, then for sufficiently large $C$ the solution achieves the maximal margin separator; if not, the solution achieves the minimum overlap solution with largest margin.

Alternatively, we can formulate the problem using a *(hinge) Loss + Penalty* criterion [4, 5]:

$$\min_{\beta_0, \beta} \sum_{i=1}^{n} [1 - y_i(\beta_0 + \beta^T x_i)]_+ + \frac{\lambda}{2}||\beta||^2. \tag{2}$$

The regularization parameter $\lambda$ in (2) corresponds to $1/C$, with $C$ in (1).

This latter formulation emphasizes the role of regularization. In many situations we have sufficient variables (e.g. gene expression arrays) to guarantee separation. We may nevertheless avoid the maximum margin separator ($\lambda \downarrow 0$), which is governed by observations on the boundary, in favor of a more regularized solution involving more observations.

The nonlinear *kernel* SVMs can be represented in this form as well. With kernel $K$ and $f(x) = \beta_0 + \sum_{i=1}^{n} \theta_i K(x, x_i)$, we solve [5]

$$\min_{\beta_0, \theta} \sum_{i=1}^{n} [1 - y_i(\beta_0 + \sum_{j=1}^{n} \theta_i K(x_i, x_j))] + \frac{\lambda}{2} \sum_{j=1}^{n} \sum_{j'=1}^{n} \theta_j \theta_{j'} K(x_j, x'_j). \tag{3}$$

Often the regularization parameter $C$ (or $\lambda$) is regarded as a genuine "nuisance". Software packages, such as the widely used *SVM$^{light}$* [6], provide default settings for $C$.

To illustrate the effect of regularization, we generated data from a pair of mixture densities, described in detail in [5, Chapter 2]. We used an SVM with a radial kernel $K(x, x') = \exp(-\gamma||x - x'||^2)$. Figure 1 shows the test error as a function of $C$ for these data, using four different values for $\gamma$. Here we see a dramatic range in the correct choice for $C$ (or $\lambda = 1/C$). When $\gamma = 5$, the most regularized model is called for; when $\gamma = 0.1$, the least regularized.

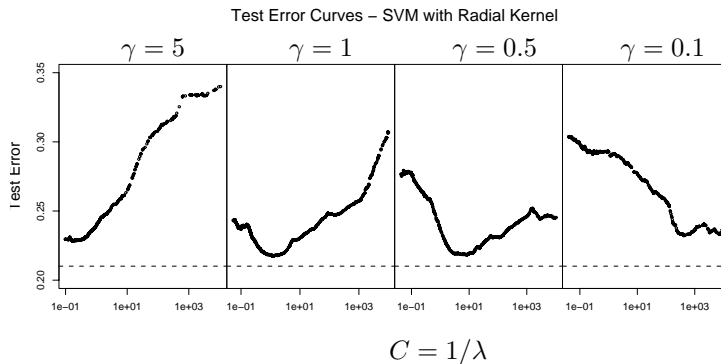

Figure 1: *Test error curves for the mixture example, using four different values for the radial kernel parameter $\gamma$.*

One of the reasons that investigators avoid extensive exploration of $C$ is the computational cost involved. In this paper we develop an algorithm which fits the *entire path* of SVM solutions $[\beta_0(C), \beta(C)]$, for all possible values of $C$, with essentially the computational cost of fitting a single model for a particular value of $C$. Our algorithm exploits the fact that the Lagrange multipliers implicit in (1) are piecewise-linear in $C$. This also means that the coefficients $\hat{\beta}(C)$ are also piecewise-linear in $C$. This is true for all SVM models, both linear and nonlinear kernel-based SVMs.

## 2    Problem Setup

We use a criterion equivalent to (1), implementing the formulation in (2):

$$\min_{\beta, \beta_0} \sum_{i=1}^{n} \xi_i + \frac{\lambda}{2} \beta^T \beta \text{ subject to } 1 - y_i f(x_i) \leq \xi_i; \ \xi_i \geq 0; \ f(x) = \beta_0 + \beta^T x. \tag{4}$$

Initially we consider only linear SVMs to get the intuitive flavor of our procedure; we then generalize to kernel SVMs.

We construct the Lagrange primal function

$$L_P: \quad \sum_{i=1}^{n} \xi_i + \frac{\lambda}{2}\beta^T\beta + \sum_{i=1}^{n}\alpha_i(1 - y_i f(x_i) - \xi_i) - \sum_{i=1}^{n}\gamma_i\xi_i \tag{5}$$

and set the derivatives to zero. This gives

$$\frac{\partial}{\partial\beta}: \quad \beta = \frac{1}{\lambda}\sum_{i=1}^{n}\alpha_i y_i x_i \tag{6}$$

$$\frac{\partial}{\partial\beta_0}: \quad \sum_{i=1}^{n}y_i\alpha_i = 0, \tag{7}$$

along with the KKT conditions

$$\alpha_i(1 - y_i f(x_i) - \xi_i) = 0 \tag{8}$$
$$\gamma_i\xi_i = 0 \tag{9}$$
$$1 - \alpha_i - \gamma_i = 0 \tag{10}$$

We see that $0 \leq \alpha_i \leq 1$, with $\alpha_i = 1$ when $\xi_i > 0$ (which is when $y_i f(x_i) < 1$). Also when $y_i f(x_i) > 1$, $\xi_i = 0$ since no cost is incurred, and $\alpha_i = 0$. When $y_i f(x_i) = 1$, $\alpha_i$ can lie between 0 and 1.

The *usual* Lagrange multipliers associated with the solution to (1) are $\alpha'_i = \alpha_i/\lambda = C\alpha_i$. We prefer our formulation here since our $\alpha_i \in [0,1]$, and this simplifies the definition of the paths we define.

We wish to find the entire solution path for all values of $\lambda \geq 0$. Our basic idea is as follows. We start with $\lambda$ large and decrease it toward zero, keeping track of all the events that occur along the way. As $\lambda$ decreases, $||\beta||$ increases, and hence the width of the margin decreases. As this width decreases, points move from being inside to outside their margins. Their corresponding $\alpha_i$ change from $\alpha_i = 1$ when they are inside their margin ($y_i f(x_i) < 1$) to $\alpha_i = 0$ when they are outside their margin ($y_i f(x_i) > 1$). By continuity, points must linger on the margin ($y_i f(x_i) = 1$) while their $\alpha_i$ decrease from 1 to 0. We will see that the $\alpha_i(\lambda)$ trajectories are piecewise-linear in $\lambda$, which affords a great computational savings: as long as we can establish the break points, all values in between can be found by simple linear interpolation. Note that points can return to the margin, after having passed through it.

It is easy to show that if the $\alpha_i(\lambda)$ are piecewise linear in $\lambda$, then both $\alpha'_i(C) = C\alpha_i(C)$ and $\beta(C)$ are piecewise linear in $C$. It turns out that $\beta_0(C)$ is also piecewise linear in $C$.

Our algorithm keeps track of the following sets:

- $\mathcal{M} = \{i : y_i f(x_i) = 1, \; 0 \leq \alpha_i \leq 1\}$, $\mathcal{M}$ for Margin
- $\mathcal{I} = \{i : y_i f(x_i) < 1, \; \alpha_i = 1\}$, $\mathcal{I}$ for Inside the margin
- $\mathcal{O} = \{i : y_i f(x_i) > 1, \; \alpha_i = 0\}$, $\mathcal{O}$ for Outside the margin

## 3    The Algorithm

Due to space restrictions, we show some details here; the rest can be found in [7].

**Initialization**

The initial conditions depend on whether the classes are balanced or not ($n_+ = n_-$). The balanced case is easier. For very large $\lambda$, $||\beta||$ is small, and the the margin is very wide,

all points are in $\mathcal{O}$, and hence $\alpha_i = 1\forall i$. From (6) this means the orientation of $\beta$ is fixed until the $\alpha_i$ change. The margin narrows as $\lambda$ decreases, but the orientation remains fixed. Because of (7), the narrowing margin must connect with an outermost member of each class simultaneously. These points are easily identified, and this establishes the first event, the first tenants of $\mathcal{M}$, and $\beta_0$.

When $n_- \neq n_+$, the setup is more complex. In order to satisfy the constraint (7), a quadratic programming algorithm is needed to obtain the initial configuration. See [7] for details.

### Kernels

The development so far has been in the original feature space. It is easy to see that the entire development carries through with "kernels" as well. In this case $f(x) = \beta_0 + g(x)$, and the only change that occurs is that (6) is changed to

$$g(x_i) = \frac{1}{\lambda} \sum_{j=1}^{n} \alpha_j y_j K(x_i, x_j), \ i = 1, \ldots, n, \tag{11}$$

or $\theta_j(\lambda) = \alpha_j y_j / \lambda$ using the notation in (3). Hereafter we will develop our algorithm for this more general kernel case.

### The Path

The algorithm hinges on the set of points $\mathcal{M}$ sitting on the margin. We consider $\mathcal{M}$ at the point that an event has occurred:

1. The initial event, which means 2 or more points start in $\mathcal{M}$, with their initial values of $\alpha \in [0, 1]$.

2. A point from $\mathcal{I}$ has just entered $\mathcal{M}$, with its value of $\alpha_i$ initially 1.

3. A point from $\mathcal{O}$ has reentered $\mathcal{M}$, with its value of $\alpha_i$ initially 0.

4. One or more points in $\mathcal{M}$ has left the set, to join either $\mathcal{O}$ or $\mathcal{I}$.

Whichever the case, for continuity reasons this set will stay stable until the next event occurs, since to pass through $\mathcal{M}$, a point's $\alpha_i$ must change from 0 to 1 or vice versa. Since all points in $\mathcal{M}$ have $y_i f(x_i) = 1$, we can establish a path for their $\alpha_i$.

We use the subscript $\ell$ to index the sets above immediately after the $\ell$th event has occurred. Suppose $|\mathcal{M}_\ell| = m$, and let $\alpha_i^\ell$, $\beta_0^\ell$ and $\lambda_\ell$ be the values of these parameters at the point of entry. Likewise $f^\ell$ is the function at this point. For convenience we define $\alpha_0 = \lambda\beta_0$, and hence $\alpha_0^\ell = \lambda_\ell \beta_0^\ell$.

Since

$$f(x) = \frac{1}{\lambda} \left( \sum_{j=1}^{n} y_j \alpha_j K(x, x_j) + \alpha_0 \right), \tag{12}$$

for $\lambda_\ell > \lambda > \lambda_{\ell+1}$ we can write

$$\begin{aligned} f(x) &= \left[ f(x) - \frac{\lambda_\ell}{\lambda} f^\ell(x) \right] + \frac{\lambda_\ell}{\lambda} f^\ell(x) \\ &= \frac{1}{\lambda} \left[ \sum_{j \in \mathcal{M}_\ell} (\alpha_j - \alpha_j^\ell) y_j K(x, x_j) + (\alpha_0 - \alpha_0^\ell) + \lambda_\ell f^\ell(x) \right]. \end{aligned} \tag{13}$$

The second line follows because all the observations in $\mathcal{I}_\ell$ have their $\alpha_i = 1$, and those in $\mathcal{O}_\ell$ have their $\alpha_i = 0$, for this range of $\lambda$. Since each of the $m$ points $x_i \in \mathcal{M}_\ell$ are to stay on the margin, we have that

$$\frac{1}{\lambda} \left[ \sum_{j \in \mathcal{M}_\ell} (\alpha_j - \alpha_j^\ell) y_i y_j K(x_i, x_j) + y_i(\alpha_0 - \alpha_0^\ell) + \lambda_\ell \right] = 1, \ \forall i \in \mathcal{M}_\ell. \quad (14)$$

Writing $\delta_j = \alpha_j^\ell - \alpha_j$, from (14) we have

$$\sum_{j \in \mathcal{M}_\ell} \delta_j y_i y_j K(x_i, x_j) + y_i \delta_0 = \lambda_\ell - \lambda, \ \forall i \in \mathcal{M}_\ell. \quad (15)$$

Furthermore, since at all times $\sum_{i=1}^n y_i \alpha_i = 0$, we have that

$$\sum_{j \in \mathcal{M}_\ell} y_j \delta_j = 0. \quad (16)$$

Equations (15) and (16) constitute $m + 1$ linear equations in $m + 1$ unknowns $\delta_j$, and can be solved. The $\delta_j$ and hence $\alpha_j$ will change linearly in $\lambda$, until the next event occurs:

$$\alpha_j = \alpha_j^\ell - (\lambda_\ell - \lambda) b_j, \ j \in \{0\} \cup \mathcal{M}_\ell. \quad (17)$$

See [7] for more precise details on solving these equations.

From (13) we have

$$f(x) = \frac{\lambda_\ell}{\lambda} \left[ f^\ell(x) - h^\ell(x) \right] + h^\ell(x), \quad (18)$$

where

$$h^\ell(x) = \sum_{j \in \mathcal{M}_\ell} y_j b_j K(x, x_j) + b_0 \quad (19)$$

Thus the function itself changes in a piecewise-inverse manner in $\lambda$.

**Finding $\lambda_{\ell+1}$**

The paths continue until one of the following events occur:

1. One of the $\alpha_i$ for $i \in \mathcal{M}_\ell$ reaches a boundary (0 or 1). For each $i$ the value of $\lambda$ for which this occurs is easily established.

2. One of the points in $\mathcal{I}^\ell$ or $\mathcal{O}^\ell$ attains $y_i f(x_i) = 1$.

By examining these conditions, we can establish the largest $\lambda < \lambda_\ell$ for which an event occurs, and hence establish $\lambda_{\ell+1}$ and update the sets.

**Termination**

In the separable case, we terminate when $\mathcal{I}$ becomes empty. At this point, all the $\xi_i$ in (4) are zero, and further movement increases the norm of $\beta$ unnecessarily.

In the non-separable case, $\lambda$ runs all the way down to zero. For this to happen without $f$ "blowing up" in (18), we must have $f^\ell - h^\ell = 0$, and hence the boundary and margins remain fixed at a point where $\sum_i \xi_i$ is as small as possible, and the margin is as wide as possible subject to this constraint.

### 3.1 Computational Complexity

At any update event $\ell$ along the path of our algorithm, the main computational burden is solving the system of equations of size $m_\ell = |\mathcal{M}_\ell|$. While this normally involves $O(m_\ell^3)$ computations, since $\mathcal{M}_{\ell+1}$ differs from $\mathcal{M}_\ell$ by typically one observation, inverse updating can reduce the computations to $O(m_\ell^2)$. The computation of $h^\ell(x_i)$ in (19) requires $O(nm_\ell)$ computations. Beyond that, several checks of cost $O(n)$ are needed to evaluate the next move.

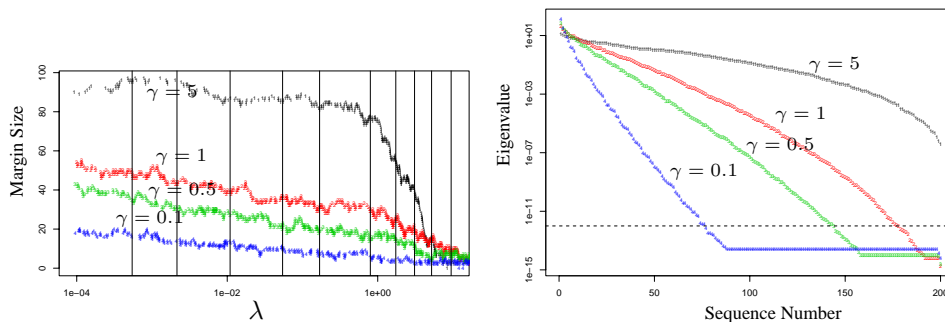

Figure 2: *[Left] The margin sizes $|\mathcal{M}_\ell|$ as a function of $\lambda$, for different values of the radial-kernel parameter $\gamma$. The vertical lines show the positions used to compare the times with* libsvm*. [Right] The eigenvalues (on the log scale) for the kernel matrices $\mathbf{K}_\gamma$ corresponding to the four values of $\gamma$. The larger eigenvalues correspond in this case to smoother eigenfunctions, the small ones to rougher. The rougher eigenfunctions get penalized exponentially more than the smoother ones. For smaller values of $\gamma$, the effective dimension of the space is truncated.*

Although we have no hard results, our experience so far suggests that the total number $\Lambda$ of moves is $O(k \min(n_+, n_-))$, for $k$ around $4-6$; hence typically some small multiple $c$ of $n$. If the average size of $\mathcal{M}_\ell$ is $m$, this suggests the total computational burden is $O(cn^2 m + nm^2)$, which is similar to that of a single SVM fit.

Our R function SvmPath computes all 632 steps in the mixture example ($n_+ = n_- = 100$, radial kernel, $\gamma = 1$) in 1.44(0.02) secs on a Pentium 4, 2Ghz Linux machine; the svm function (using the optimized code libsvm, from the R library e1071) takes 9.28(0.06) seconds to compute the solution at 10 points along the path. Hence it takes our procedure about 50% more time to compute the entire path, than it costs libsvm to compute a typical single solution.

## 4  Mixture simulation continued

The $\lambda_\ell$ in Figure 1 are the *entire* collection of change points as described in Section 3. We were at first surprised to discover that not all these sequences achieved zero training errors on the 200 training data points, at their least regularized fit. In fact the minimal training errors, and the corresponding values for $\gamma$ are summarized in Table 1. It is sometimes argued that the implicit feature space is "infinite dimensional" for this kernel, which suggests that perfect separation is always possible. The last row of the table shows the effective rank of the $200 \times 200$ kernel *Gram* matrix $\mathbf{K}$ (which we defined to be the number of singular values greater than $10^{-12}$). In general a full rank $\mathbf{K}$ is required to achieve perfect separation. This rank-deficiency of the Gram matrix has been noted by a number of other authors.

This emphasizes the fact that not all features in the feature map implied by $K$ are of equal

| $\gamma$ | 5 | 1 | 0.5 | 0.1 |
|---|---|---|---|---|
| Training Errors | 0 | 12 | 21 | 33 |
| Effective Rank | 200 | 177 | 143 | 76 |

Table 1: *The number of minimal training errors for different values of the radial kernel scale parameter $\gamma$, for the mixture simulation example. Also shown is the effective rank of the $200 \times 200$ Gram matrix $\mathbf{K}_\gamma$.*

stature; many of them are shrunk way down to zero. Rephrasing, the regularization in (3) penalizes unit-norm features by the inverse of their eigenvalues, which effectively annihilates some, depending on $\gamma$. Small $\gamma$ implies wide, flat kernels, and a suppression of wiggly, "rough" functions.

Writing (3) in matrix form,

$$\min_{\beta_0, \boldsymbol{\theta}} L[\mathbf{y}, \mathbf{K}\boldsymbol{\theta}] + \frac{\lambda}{2}\boldsymbol{\theta}^T \mathbf{K}\boldsymbol{\theta}, \tag{20}$$

we reparametrize using the eigen-decomposition of $\mathbf{K} = \mathbf{U}\mathbf{D}\mathbf{U}^T$. Let $\mathbf{K}\boldsymbol{\theta} = \mathbf{U}\boldsymbol{\theta}^*$ where $\boldsymbol{\theta}^* = \mathbf{D}\mathbf{U}^T \theta$. Then (20) becomes

$$\min_{\beta_0, \boldsymbol{\theta}^*} L[\mathbf{y}, \mathbf{U}\boldsymbol{\theta}^*] + \frac{\lambda}{2}\boldsymbol{\theta}^{*T} \mathbf{D}^{-1}\boldsymbol{\theta}^*. \tag{21}$$

Now the columns of $\mathbf{U}$ are unit-norm basis functions (in $\mathbb{R}^2$) spanning the column space of $\mathbf{K}$; from (21) we see that those members corresponding to near-zero eigenvalues (the elements of the diagonal matrix $\mathbf{D}$) get heavily penalized and hence ignored. Figure 2 shows the elements of $\mathbf{D}$ for the four values of $\gamma$.

## 5  Discussion

Our work on the SVM path algorithm was inspired by early work on exact path algorithms in other settings. "Least Angle Regression" [8] show that the coefficient path for the sequence of "lasso" coefficients is piecewise linear. The lasso uses a quadratic criterion, with an $L_1$ constraint. In fact, any model with an $L_1$ constraint and a quadratic, piecewise quadratic, piecewise linear, or mixed quadratic and linear loss function, will have piecewise linear coefficient paths, which can be calculated exactly and efficiently for all values of $\lambda$ [9]. This includes the $L_1$ SVM [10].

The SVM model has a quadratic constraint and a piecewise linear ("hinge") loss function. This leads to a piecewise linear path in the dual space, hence the Lagrange coefficients $\alpha_i$ are piecewise linear.

Of course, quadratic criterion + quadratic constraints also lead to exact path solutions, as in the classic ridge regression case, since a closed form solution is obtained via the SVD.

The general techniques employed in this paper are known as parametric programming in convex optimization. After completing this work, it was brought to our attention that [11] reported on the picewise-linear nature of the lagrange multipliers, although they did not develop the path algorithm. [12, 13] employ techniques similar to ours in incremental learning for SVMs. These authors do not construct exact paths as we do, but rather focus on updating and downdating the solutions as more (or less) data arises. [14] allow for updating the parameters as well, but again do not construct entire solution paths.

The `SvmPath` has been implemented in the `R` computing environment, and is available from the R website.

**Acknowledgements**

The authors thank Jerome Friedman for helpful discussions, and Mee-Young Park for assisting with some of the computations. Trevor Hastie was partially supported by grant DMS-0204162 from the National Science Foundation, and grant RO1-EB0011988-08 from the National Institutes of Health.

# References

[1] B. Boser, I. Guyon, and V. Vapnik. A training algorithm for optimal margin classifiers. In *Proceedings of COLT II*, Philadelphia, PA, 1992.

[2] C. Cortes and V. Vapnik. Support vector networks. *Machine Learning*, 20:1–25, 1995.

[3] Bernard Schölkopf and Alex Smola. *Learning with Kernels: Support Vector Machines, Regularization, Optimization, and Beyond (Adaptive Computation and Machine Learning)*. MIT Press, 2001.

[4] G. Wahba, Y. Lin, and H. Zhang. Gacv for support vector machines. In A.J. Smola, P.L. Bartlett, B. Schölkopf, and D. Schuurmans, editors, *Advances in Large Margin Classifiers*, pages 297–311, Cambridge, MA, 2000. MIT Press.

[5] T. Hastie, R. Tibshirani, and J. Friedman. *The Elements of Statistical Learning; Data mining, Inference and Prediction*. Springer Verlag, New York, 2001.

[6] Thorsten Joachims. *Practical Advances in Kernel Methods — Support Vector Learning*, chapter Making large scale SVM learning practical. MIT Press, 1999. see `http://svmlight.joachims.org`.

[7] Trevor Hastie, Saharon Rosset, Robert Tibshirani, and Ji Zhu. The entire regularization path for the support vector machine. *Journal of Machine Learning Research*, (5):1391–1415, 2004.

[8] B. Efron, T. Hastie, I. Johnstone, and R.. Tibshirani. Least angle regression. Technical report, Stanford University, 2002.

[9] Saharon Rosset and Ji Zhu. Piecewise linear regularized solution paths. Technical report, Stanford University, 2003. `http://www-stat.stanford.edu/~saharon/papers/piecewise.ps`.

[10] Ji Zhu, Saharon Rosset, Trevor Hastie, and Robert Tibshirani. L1 norm support vector machines. Technical report, Stanford University, 2003.

[11] Massimiliano Pontil and Alessandro Verri. Properties of support vector machines. *Neural Comput.*, 10(4):955–974, 1998.

[12] Shai Fine and Katya Scheinberg. Incas: An incremental active set method for svm. Technical report, IBM Research Labs, Haifa, 2002.

[13] G. Cauwenberghs and T. Poggio. Incremental and decremental support vector machine learning. In *Advances in Neural Information Processing Systems (NIPS*2000)*, volume 13. MIT Press, Cambridge, MA, 2001.

[14] Christopher Diehl and Gert Cauwenberghs. Svm incremental learning, adaptation and optimization. In *Proceedings of the 2003 International Joint Conference on Neural Networks*, pages 2685–2690, 2003. Special series on Incremental Learning.
